# Using Analytic QP and Sparseness to Speed Training of Support Vector Machines

**John C. Platt**
Microsoft Research
1 Microsoft Way
Redmond, WA 98052
jplatt@microsoft.com

## Abstract

Training a Support Vector Machine (SVM) requires the solution of a very large quadratic programming (QP) problem. This paper proposes an algorithm for training SVMs: *Sequential Minimal Optimization*, or *SMO*. SMO breaks the large QP problem into a series of smallest possible QP problems which are analytically solvable. Thus, SMO does not require a numerical QP library. SMO's computation time is dominated by evaluation of the kernel, hence kernel optimizations substantially quicken SMO. For the MNIST database, SMO is 1.7 times as fast as PCG chunking; while for the UCI Adult database and linear SVMs, SMO can be 1500 times faster than the PCG chunking algorithm.

## 1 INTRODUCTION

In the last few years, there has been a surge of interest in Support Vector Machines (SVMs) [1]. SVMs have empirically been shown to give good generalization performance on a wide variety of problems. However, the use of SVMs is still limited to a small group of researchers. One possible reason is that training algorithms for SVMs are slow, especially for large problems. Another explanation is that SVM training algorithms are complex, subtle, and sometimes difficult to implement. This paper describes a new SVM learning algorithm that is easy to implement, often faster, and has better scaling properties than the standard SVM training algorithm. The new SVM learning algorithm is called Sequential Minimal Optimization (or SMO).

### 1.1 OVERVIEW OF SUPPORT VECTOR MACHINES

A general non-linear SVM can be expressed as

$$u = \sum_i \alpha_i y_i K(\vec{x}_i, \vec{x}) - b \qquad (1)$$

where $u$ is the output of the SVM, $K$ is a kernel function which measures the similarity of a stored training example $\vec{x}_i$ to the input $\vec{x}$, $y_i \in \{-1, +1\}$ is the desired output of the classifier, $b$ is a threshold, and $\alpha_i$ are weights which blend the different kernels [1]. For linear SVMs, the kernel function $K$ is linear, hence equation (1) can be expressed as

$$u = \vec{w} \cdot \vec{x} - b \tag{2}$$

where $\vec{w} = \sum_i \alpha_i y_i \vec{x}_i$.

Training of an SVM consists of finding the $\alpha_i$. The training is expressed as a minimization of a dual quadratic form:

$$\min_{\vec{\alpha}} \Psi(\alpha) = \min_{\vec{\alpha}} \frac{1}{2} \sum_{i=1}^{N} \sum_{j=1}^{N} y_i y_j K(\vec{x}_i, \vec{x}_j) \alpha_i \alpha_j - \sum_{i=1}^{N} \alpha_i, \tag{3}$$

subject to box constraints,

$$0 \leq \alpha_i \leq C, \quad \forall i, \tag{4}$$

and one linear equality constraint

$$\sum_{i=1}^{N} y_i \alpha_i = 0. \tag{5}$$

The $\alpha_i$ are Lagrange multipliers of a primal quadratic programming (QP) problem: there is a one-to-one correspondence between each $\alpha_i$ and each training example $\vec{x}_i$.

Equations (3–5) form a QP problem that the SMO algorithm will solve. The SMO algorithm will terminate when all of the Karush-Kuhn-Tucker (KKT) optimality conditions of the QP problem are fulfilled. These KKT conditions are particularly simple:

$$\alpha_i = 0 \Rightarrow y_i u_i \geq 1, \quad 0 < \alpha_i < C \Rightarrow y_i u_i = 1, \quad \alpha_i = C \Rightarrow y_i u_i \leq 1, \tag{6}$$

where $u_i$ is the output of the SVM for the $i$th training example.

## 1.2   PREVIOUS METHODS FOR TRAINING SUPPORT VECTOR MACHINES

Due to its immense size, the QP problem that arises from SVMs cannot be easily solved via standard QP techniques. The quadratic form in (3) involves a Hessian matrix of dimension equal to the number of training examples. This matrix cannot be fit into 128 Megabytes if there are more than 4000 training examples.

Vapnik [9] describes a method to solve the SVM QP, which has since been known as "chunking." Chunking relies on the fact that removing training examples with $\alpha_i = 0$ does not change the solution. Chunking thus breaks down the large QP problem into a series of smaller QP sub-problems, whose object is to identify the training examples with non-zero $\alpha_i$. Every QP sub-problem updates the subset of the $\alpha_i$ that are associated with the sub-problem, while leaving the rest of the $\alpha_i$ unchanged. The QP sub-problem consists of every non-zero $\alpha_i$ from the previous sub-problem combined with the $M$ worst examples that violate the KKT conditions (6), for some $M$ [1]. At the last step, the entire set of non-zero $\alpha_i$ has been identified, hence the last step solves the entire QP problem.

Chunking reduces the dimension of the matrix from the number of training examples to approximately the number of non-zero $\alpha_i$. If standard QP techniques are used, chunking cannot handle large-scale training problems, because even this reduced matrix cannot fit into memory. Kaufman [3] has described a QP algorithm that does not require the storage of the entire Hessian.

The decomposition technique [6] is similar to chunking: decomposition breaks the large QP problem into smaller QP sub-problems. However, Osuna et al. [6] suggest keeping a

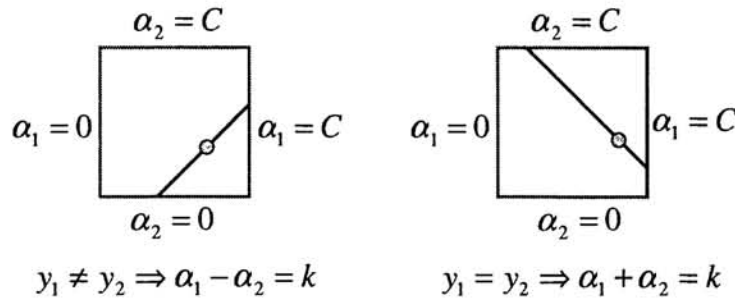

$$y_1 \neq y_2 \Rightarrow \alpha_1 - \alpha_2 = k \qquad y_1 = y_2 \Rightarrow \alpha_1 + \alpha_2 = k$$

Figure 1: The Lagrange multipliers $\alpha_1$ and $\alpha_2$ must fulfill all of the constraints of the full problem. The inequality constraints cause the Lagrange multipliers to lie in the box. The linear equality constraint causes them to lie on a diagonal line.

fixed size matrix for every sub-problem, deleting some examples and adding others which violate the KKT conditions. Using a fixed-size matrix allows SVMs to be trained on very large training sets. Joachims [2] suggests adding and subtracting examples according to heuristics for rapid convergence. However, until SMO, decomposition required the use of a numerical QP library, which can be costly or slow.

## 2 SEQUENTIAL MINIMAL OPTIMIZATION

Sequential Minimal Optimization quickly solves the SVM QP problem without using numerical QP optimization steps at all. SMO decomposes the overall QP problem into fixed-size QP sub-problems, similar to the decomposition method [7].

Unlike previous methods, however, SMO chooses to solve the smallest possible optimization problem at each step. For the standard SVM, the smallest possible optimization problem involves two elements of $\vec{\alpha}$ because the $\vec{\alpha}$ must obey one linear equality constraint. At each step, SMO chooses two $\alpha_i$ to jointly optimize, finds the optimal values for these $\alpha_i$, and updates the SVM to reflect these new values.

The advantage of SMO lies in the fact that solving for two $\alpha_i$ can be done analytically. Thus, numerical QP optimization is avoided entirely. The inner loop of the algorithm can be expressed in a short amount of C code, rather than invoking an entire QP library routine.

By avoiding numerical QP, the computation time is shifted from QP to kernel evaluation. Kernel evaluation time can be dramatically reduced in certain common situations, e.g., when a linear SVM is used, or when the input data is sparse (mostly zero). The result of kernel evaluations can also be cached in memory [1].

There are two components to SMO: an analytic method for solving for the two $\alpha_i$, and a heuristic for choosing which multipliers to optimize. Pseudo-code for the SMO algorithm can be found in [8, 7], along with the relationship to other optimization and machine learning algorithms.

### 2.1 SOLVING FOR TWO LAGRANGE MULTIPLIERS

To solve for the two Lagrange multipliers $\alpha_1$ and $\alpha_2$, SMO first computes the constraints on these multipliers and then solves for the constrained minimum. For convenience, all quantities that refer to the first multiplier will have a subscript 1, while all quantities that refer to the second multiplier will have a subscript 2. Because there are only two multipliers,

the constraints can easily be displayed in two dimensions (see figure 1). The constrained minimum of the objective function must lie on a diagonal line segment.

The ends of the diagonal line segment can be expressed quite simply in terms of $\alpha_2$. Let $s = y_1 y_2$. The following bounds apply to $\alpha_2$:

$$L = \max(0, \alpha_2 + s\alpha_1 - \frac{1}{2}(s+1)C), \qquad H = \min(C, \alpha_2 + s\alpha_1 - \frac{1}{2}(s-1)C). \quad (7)$$

Under normal circumstances, the objective function is positive definite, and there is a minimum along the direction of the linear equality constraint. In this case, SMO computes the minimum along the direction of the linear equality constraint:

$$\alpha_2^{\text{new}} = \alpha_2 + \frac{y_2(E_1 - E_2)}{K(\vec{x}_1, \vec{x}_1) + K(\vec{x}_2, \vec{x}_2) - 2K(\vec{x}_1, \vec{x}_2)}, \quad (8)$$

where $E_i = u_i - y_i$ is the error on the $i$th training example. As a next step, the constrained minimum is found by clipping $\alpha_2^{\text{new}}$ into the interval $[L, H]$. The value of $\alpha_1$ is then computed from the new, clipped, $\alpha_2$:

$$\alpha_1^{\text{new}} = \alpha_1 + s(\alpha_2 - \alpha_2^{\text{new,clipped}}). \quad (9)$$

For both linear and non-linear SVMs, the threshold $b$ is re-computed after each step, so that the KKT conditions are fulfilled for both optimized examples.

## 2.2  HEURISTICS FOR CHOOSING WHICH MULTIPLIERS TO OPTIMIZE

In order to speed convergence, SMO uses heuristics to choose which two Lagrange multipliers to jointly optimize.

There are two separate choice heuristics: one for $\alpha_1$ and one for $\alpha_2$. The choice of $\alpha_1$ provides the outer loop of the SMO algorithm. If an example is found to violate the KKT conditions by the outer loop, it is eligible for optimization. The outer loop alternates single passes through the entire training set with multiple passes through the non-bound $\alpha_i$ ($\alpha_i \neq \{0, C\}$). The multiple passes terminate when all of the non-bound examples obey the KKT conditions within $\epsilon$. The entire SMO algorithm terminates when the entire training set obeys the KKT conditions within $\epsilon$. Typically, $\epsilon = 10^{-3}$.

The first choice heuristic concentrates the CPU time on the examples that are most likely to violate the KKT conditions, i.e., the non-bound subset. As the SMO algorithm progresses, $\alpha_i$ that are at the bounds are likely to stay at the bounds, while $\alpha_i$ that are not at the bounds will move as other examples are optimized.

As a further optimization, SMO uses the shrinking heuristic proposed in [2]. After the pass through the entire training set, shrinking finds examples which fulfill the KKT conditions more than the worst example failed the KKT conditions. Further passes through the training set ignore these fulfilled conditions until a final pass at the end of training, which ensures that every example fulfills its KKT condition.

Once an $\alpha_1$ is chosen, SMO chooses an $\alpha_2$ to maximize the size of the step taken during joint optimization. SMO approximates the step size by the absolute value of the numerator in equation (8): $|E_1 - E_2|$. SMO keeps a cached error value $E$ for every non-bound example in the training set and then chooses an error to approximately maximize the step size. If $E_1$ is positive, SMO chooses an example with minimum error $E_2$. If $E_1$ is negative, SMO chooses an example with maximum error $E_2$.

## 2.3  KERNEL OPTIMIZATIONS

Because the computation time for SMO is dominated by kernel evaluations, SMO can be accelerated by optimizing these kernel evaluations. Utilizing sparse inputs is a generally

| Experiment | Kernel | Sparse Inputs Used | Kernel Caching Used | Training Set Size | Number of Support Vectors | C | % Sparse Inputs |
|---|---|---|---|---|---|---|---|
| AdultLin | Linear | Y | mix | 11221 | 4158 | 0.05 | 89 |
| AdultLinD | Linear | N | mix | 11221 | 4158 | 0.05 | 0 |
| WebLin | Linear | Y | mix | 49749 | 1723 | 1 | 96 |
| WebLinD | Linear | N | mix | 49749 | 1723 | 1 | 0 |
| AdultGaussK | Gaussian | Y | Y | 11221 | 4206 | 1 | 89 |
| AdultGauss | Gaussian | Y | N | 11221 | 4206 | 1 | 89 |
| AdultGaussKD | Gaussian | N | Y | 11221 | 4206 | 1 | 0 |
| AdultGaussD | Gaussian | N | N | 11221 | 4206 | 1 | 0 |
| WebGaussK | Gaussian | Y | Y | 49749 | 4484 | 5 | 96 |
| WebGauss | Gaussian | Y | N | 49749 | 4484 | 5 | 96 |
| WebGaussKD | Gaussian | N | Y | 49749 | 4484 | 5 | 0 |
| WebGaussD | Gaussian | N | N | 49749 | 4484 | 5 | 0 |
| MNIST | Polynom. | Y | N | 60000 | 3450 | 100 | 81 |

Table 1: Parameters for various experiments

applicable kernel optimization. For commonly-used kernels, equations (1) and (2) can be dramatically sped up by exploiting the sparseness of the input. For example, a Gaussian kernel can be expressed as an exponential of a linear combination of sparse dot products. Sparsely storing the training set also achieves substantial reduction in memory consumption.

To compute a linear SVM, only a single weight vector needs to be stored, rather than all of the training examples that correspond to non-zero $\alpha_i$. If the QP sub-problem succeeds, the stored weight vector is updated to reflect the new $\alpha_i$ values.

## 3  BENCHMARKING SMO

The SMO algorithm is tested against the standard chunking algorithm and against the decomposition method on a series of benchmarks. Both SMO and chunking are written in C++, using Microsoft's Visual C++ 6.0 compiler. Joachims' package SVM[light] (version 2.01) with a default working set size of 10 is used to test the decomposition method. The CPU time of all algorithms are measured on an unloaded 266 MHz Pentium II processor running Windows NT 4.

The chunking algorithm uses the projected conjugate gradient algorithm as its QP solver, as suggested by Burges [1]. All algorithms use sparse dot product code and kernel caching, as appropriate [1, 2]. Both SMO and chunking share folded linear SVM code.

The SMO algorithm is tested on three real-world data sets. The results of the experiments are shown in Tables 1 and 2. Further tests on artificial data sets can be found in [8, 7].

The first test set is the UCI Adult data set [5]. The SVM is given 14 attributes of a census form of a household and asked to predict whether that household has an income greater than $50,000. Out of the 14 attributes, eight are categorical and six are continuous. The six continuous attributes are discretized into quintiles, yielding a total of 123 binary attributes.

The second test set is text categorization: classifying whether a web page belongs to a category or not. Each web page is represented as 300 sparse binary keywords attributes.

The third test set is the MNIST database of handwritten digits, from AT&T Research Labs [4]. One classifier of MNIST, class 8, is trained. The inputs are 784-dimensional

| Experiment | SMO Time (sec) | SVM<sup>light</sup> Time (sec) | Chunking Time (sec) | SMO Scaling Exponent | SVM<sup>light</sup> Scaling Exponent | Chunking Scaling Exponent |
|---|---|---|---|---|---|---|
| AdultLin | **13.7** | 217.9 | 20711.3 | **1.8** | 2.1 | 3.1 |
| AdultLinD | **21.9** | n/a | 21141.1 | **1.0** | n/a | 3.0 |
| WebLin | **339.9** | 3980.8 | 17164.7 | **1.6** | 2.2 | 2.5 |
| WebLinD | **4589.1** | n/a | 17332.8 | **1.5** | n/a | 2.5 |
| AdultGaussK | 442.4 | **284.7** | 11910.6 | 2.0 | 2.0 | 2.9 |
| AdultGauss | **523.3** | 737.5 | n/a | 2.0 | 2.0 | n/a |
| AdultGaussKD | **1433.0** | n/a | 14740.4 | **2.5** | n/a | 2.8 |
| AdultGaussD | 1810.2 | n/a | n/a | 2.0 | n/a | n/a |
| WebGaussK | **2477.9** | 2949.5 | 23877.6 | **1.6** | 2.0 | 2.0 |
| WebGauss | **2538.0** | 6923.5 | n/a | **1.6** | 1.8 | n/a |
| WebGaussKD | **23365.3** | n/a | 50371.9 | 2.6 | n/a | **2.0** |
| WebGaussD | 24758.0 | n/a | n/a | 1.6 | n/a | n/a |
| MNIST | **19387.9** | 38452.3 | 33109.0 | n/a | n/a | n/a |

Table 2: Timings of algorithms on various data sets.

non-binary vectors and are stored as sparse vectors. A fifth-order polynomial kernel is used to match the AT&T accuracy results.

The Adult set and the Web set are trained both with linear SVMs and Gaussian SVMs with variance of 10. For the Adult and Web data sets, the $C$ parameter is chosen to optimize accuracy on a validation set. Experiments on the Adult and Web sets are performed with and without sparse inputs and with and without kernel caching, in order to determine the effect these kernel optimizations have on computation time. When a kernel cache is used, the cache size for SMO and SVM<sup>light</sup> is 40 megabytes. The chunking algorithm always uses kernel caching: matrix values from the previous QP step are re-used. For the linear experiments, SMO does not use kernel caching, while SVM<sup>light</sup> does.

In Table 2, the scaling of each algorithm is measured as a function of the training set size, which is varied by taking random nested subsets of the full training set. A line is fitted to the log of the training time versus the log of the set size. The slope of the line is an empirical scaling exponent.

## 4   CONCLUSIONS

As can be seen in Table 2, standard PCG chunking is slower than SMO for the data sets shown, even for dense inputs. Decomposition and SMO have the advantage, over standard PCG chunking, of ignoring the examples whose Lagrange multipliers are at $C$. This advantage is reflected in the scaling exponents for PCG chunking versus SMO and SVM<sup>light</sup>. PCG chunking can be altered to have a similar property [3]. Notice that PCG chunking uses the same sparse dot product code and linear SVM folding code as SMO. However, these optimizations do not speed up PCG chunking due to the overhead of numerically solving large QP sub-problems.

SMO and SVM<sup>light</sup> are similar: they decompose the large QP problem into very small QP sub-problems. SMO decomposes into even smaller sub-problems: it uses analytical solutions of two-dimensional sub-problems, while SVM<sup>light</sup> uses numerical QP to solve 10-dimensional sub-problems. The difference in timings between the two methods is partly due to the numerical QP overhead, but mostly due to the difference in heuristics and kernel optimizations. For example, SMO is faster than SVM<sup>light</sup> by an order of magnitude on

linear problems, due to linear SVM folding. However, SVM$^{\text{light}}$ can also potentially use linear SVM folding. In these experiments, SMO uses a very simple least-recently-used kernel cache of Hessian rows, while SVM$^{\text{light}}$ uses a more complex kernel cache and modifies its heuristics to utilize the kernel effectively [2]. Therefore, SMO does not benefit from the kernel cache at the largest problem sizes, while SVM$^{\text{light}}$ speeds up by a factor of 2.5 .

Utilizing sparseness to compute kernels yields a large advantage for SMO due to the lack of heavy numerical QP overhead. For the sparse data sets shown, SMO can speed up by a factor of between 3 and 13, while PCG chunking only obtained a maximum speed up of 2.1 times.

The MNIST experiments were performed without a kernel cache, because the MNIST data set takes up most of the memory of the benchmark machine. Due to sparse inputs, SMO is a factor of 1.7 faster than PCG chunking, even though none of the Lagrange multipliers are at $C$. On a machine with more memory, SVM$^{\text{light}}$ would be as fast or faster than SMO for MNIST, due to kernel caching.

In summary, SMO is a simple method for training support vector machines which does not require a numerical QP library. Because its CPU time is dominated by kernel evaluation, SMO can be dramatically quickened by the use of kernel optimizations, such as linear SVM folding and sparse dot products. SMO can be anywhere from 1.7 to 1500 times faster than the standard PCG chunking algorithm, depending on the data set.

## Acknowledgements

Thanks to Chris Burges for running data sets through his projected conjugate gradient code and for various helpful suggestions.

## References

[1] C. J. C. Burges. A tutorial on support vector machines for pattern recognition. *Data Mining and Knowledge Discovery*, 2(2), 1998.

[2] T. Joachims. Making large-scale SVM learning practical. In B. Schölkopf, C. J. C. Burges, and A. J. Smola, editors, *Advances in Kernel Methods — Support Vector Learning*, pages 169–184. MIT Press, 1998.

[3] L. Kaufman. Solving the quadratic programming problem arising in support vector classification. In B. Schölkopf, C. J. C. Burges, and A. J. Smola, editors, *Advances in Kernel Methods — Support Vector Learning*, pages 147–168. MIT Press, 1998.

[4] Y. LeCun. MNIST handwritten digit database. Available on the web at http://www.research.att.com/˜ yann/ocr/mnist/.

[5] C. J. Merz and P. M. Murphy. UCI repository of machine learning databases, 1998. [http://www.ics.uci.edu/~mlearn/MLRepository.html]. Irvine, CA: University of California, Department of Information and Computer Science.

[6] E. Osuna, R. Freund, and F. Girosi. Improved training algorithm for support vector machines. In *Proc. IEEE Neural Networks in Signal Processing '97*, 1997.

[7] J. C. Platt. Fast training of SVMs using sequential minimal optimization. In B. Schölkopf, C. J. C. Burges, and A. J. Smola, editors, *Advances in Kernel Methods — Support Vector Learning*, pages 185–208. MIT Press, 1998.

[8] J. C. Platt. Sequential minimal optimization: A fast algorithm for training support vector machines. Technical Report MSR–TR–98–14, Microsoft Research, 1998. Available at http://www.research.microsoft.com/˜ jplatt/smo.html.

[9] V. Vapnik. *Estimation of Dependences Based on Empirical Data*. Springer-Verlag, 1982.
